# Prodding the ROC Curve: Constrained Optimization of Classifier Performance

**Michael C. Mozer**[*+]**, Robert Dodier**[*]**, Michael D. Colagrosso**[*+]**,**
**César Guerra-Salcedo**[*]**, Richard Wolniewicz**[*]

*\* Advanced Technology Group*  *+ Department of Computer Science*
*Athene Software*  *University of Colorado*
*2060 Broadway*  *Campus Box 430*
*Boulder, CO 80302*  *Boulder, CO 80309*

## Abstract

When designing a two-alternative classifier, one ordinarily aims to maximize the classifier's ability to discriminate between members of the two classes. We describe a situation in a real-world business application of machine-learning prediction in which an additional constraint is placed on the nature of the solution: that the classifier achieve a specified correct acceptance or correct rejection rate (i.e., that it achieve a fixed accuracy on members of one class or the other). Our domain is predicting *churn* in the telecommunications industry. Churn refers to customers who switch from one service provider to another. We propose four algorithms for training a classifier subject to this domain constraint, and present results showing that each algorithm yields a reliable improvement in performance. Although the improvement is modest in magnitude, it is nonetheless impressive given the difficulty of the problem and the financial return that it achieves to the service provider.

When designing a classifier, one must specify an objective measure by which the classifier's performance is to be evaluated. One simple objective measure is to minimize the number of misclassifications. If the cost of a classification error depends on the target and/or response class, one might utilize a risk-minimization framework to reduce the expected loss. A more general approach is to maximize the classifier's ability to discriminate one class from another class (e.g., Chang & Lippmann, 1994).

An *ROC curve* (Green & Swets, 1966) can be used to visualize the discriminative performance of a two-alternative classifier that outputs class posteriors. To explain the ROC curve, a classifier can be thought of as making a positive/negative judgement as to whether an input is a member of some class. Two different accuracy measures can be obtained from the classifier: the accuracy of correctly identifying an input as a member of the class (a *correct acceptance* or *CA*), and the accuracy of correctly identifying an input as a nonmember of the class (a *correct rejection* or *CR*). To evaluate the CA and CR rates, it is necessary to pick a threshold above which the classifier's probability estimate is interpreted as an "accept," and below which is interpreted as a "reject"—call this the *criterion*. The ROC curve plots CA against CR rates for various criteria (Figure 1a). Note that as the threshold is lowered, the CA rate increases and the CR rate decreases. For a criterion of 1, the CA rate approaches 0 and the CR rate 1; for a criterion of 0, the CA rate approaches 1

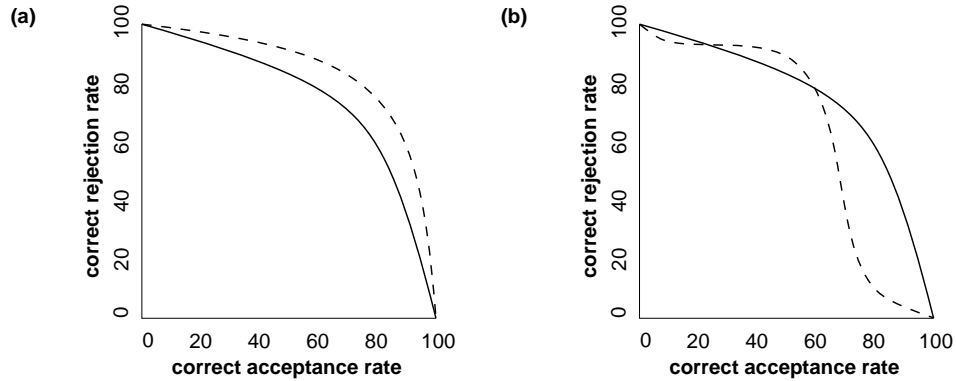

**FIGURE 1. (a) two ROC curves reflecting discrimination performance; the dashed curve indicates better performance. (b) two plausible ROC curves, neither of which is clearly superior to the other.**

and the CR rate 0. Thus, the ROC curve is anchored at (0,1) and (1,0), and is monotonically nonincreasing. The degree to which the curve is bowed reflects the discriminative ability of the classifier. The dashed curve in Figure 1a is therefore a better classifier than the solid curve.

The degree to which the curve is bowed can be quantified by various measures such as the area under the ROC curve or $d'$, the distance between the positive and negative distributions. However, training a classifier to maximize either the ROC area or d' often yields the same result as training a classifier to estimate posterior class probabilities, or equivalently, to minimize the mean squared error (e.g., Frederick & Floyd, 1998). The ROC area and d' scores are useful, however, because they reflect a classifier's intrinsic ability to discriminate between two classes, regardless of how the decision criterion is set. That is, each point on an ROC curve indicates one possible CA/CR trade off the classifier can achieve, and that trade off is determined by the criterion. But changing the criterion does not change the classifier's intrinsic ability to discriminate.

Generally, one seeks to optimize the discrimination performance of a classifier. However, we are working in a domain where overall discrimination performance is not as critical as performance at a particular point on the ROC curve, and we are not interested in the remainder of the ROC curve. To gain an intuition as to why this goal should be feasible, consider Figure 1b. Both the solid and dashed curves are valid ROC curves, because they satisfy the monotonicity constraint: as the criterion is lowered, the CA rate does not decrease and the CR rate does not increase. Although the bow shape of the solid curve is typical, it is not mandatory; the precise shape of the curve depends on the nature of the classifier and the nature of the domain. Thus, it is conceivable that a classifier could produce a curve like the dashed one. The dashed curve indicates better performance when the CA rate is around 50%, but worse performance when the CA rate is much lower or higher than 50%. Consequently, if our goal is to maximize the CR rate subject to the constraint that the CA rate is around 50%, or to maximize the CA rate subject to the constraint that the CR rate is around 90%, the dashed curve is superior to the solid curve. One can imagine that better performance can be obtained along some stretches of the curve by sacrificing performance along other stretches of the curve. Note that obtaining a result such as the dashed curve requires a nonstandard training algorithm, as the discrimination performance as measured by the ROC area is worse for the dashed curve than for the solid curve.

In this paper, we propose and evaluate four algorithms for optimizing performance in a certain region of the ROC curve. To begin, we explain the domain we are concerned with and why focusing on a certain region of the ROC curve is important in this domain.

# 1 OUR DOMAIN

Athene Software focuses on predicting and managing subscriber *churn* in the telecommunications industry (Mozer, Wolniewicz, Grimes, Johnson, & Kaushansky, 2000). "Churn" refers to the loss of subscribers who switch from one company to the other. Churn is a significant problem for wireless, long distance, and internet service providers. For example, in the wireless industry, domestic monthly churn rates are 2–3% of the customer base. Consequently, service providers are highly motivated to identify subscribers who are dissatisfied with their service and offer them incentives to prevent churn.

We use techniques from statistical machine learning—primarily neural networks and ensemble methods—to estimate the probability that an individual subscriber will churn in the near future. The prediction of churn is based on various sources of information about a subscriber, including: call detail records (date, time, duration, and location of each call, and whether call was dropped due to lack of coverage or available bandwidth), financial information appearing on a subscriber's bill (monthly base fee, additional charges for roaming and usage beyond monthly prepaid limit), complaints to the customer service department and their resolution, information from the initial application for service (contract details, rate plan, handset type, credit report), market information (e.g., rate plans offered by the service provider and its competitors), and demographic data.

Churn prediction is an extremely difficult problem for several reasons. First, the business environment is highly nonstationary; models trained on data from a certain time period perform far better with hold-out examples from that same time period than examples drawn from successive time periods. Second, features available for prediction are only weakly related to churn; when computing mutual information between individual features and churn, the greatest value we typically encounter is .01 bits. Third, information critical to predicting subscriber behavior, such as quality of service, is often unavailable.

Obtaining accurate churn predictions is only part of the challenge of subscriber retention. Subscribers who are likely to churn must be contacted by a *call center* and offered some incentive to remain with the service provider. In a mathematically principled business scenario, one would frame the challenge as maximizing profitability to a service provider, and making the decision about whether to contact a subscriber and what incentive to offer would be based on the expected utility of offering versus not offering an incentive. However, business practices complicate the scenario and place some unique constraints on predictive models. First, call centers are operated by a staff of customer service representatives who can contact subscribers at a fixed rate; consequently, our models cannot advise contacting 50,000 subscribers one week, and 50 the next. Second, internal business strategies at the service providers constrain the minimum acceptable CA or CR rates (above and beyond the goal of maximizing profitability). Third, contracts that Athene makes with service providers will occasionally call for achieving a specific target CA and CR rate. These three practical issues pose formal problems which, to the best of our knowledge, have not been addressed by the machine learning community.

The formal problems can be stated in various ways, including: (1) maximize the CA rate, subject to the constraint that a fixed percentage of the subscriber base is identified as potential churners, (2) optimize the CR rate, subject to the constraint that the CA rate should be $\alpha_{CA}$, (3) optimize the CA rate, subject to the constraint that the CR rate should be $\alpha_{CR}$, and finally—what marketing executives really want—(4) design a classifier that has a CA rate of $\alpha_{CA}$ and a CR rate of $\alpha_{CR}$. Problem (1) sounds somewhat different than problems (2) or (3), but it can be expressed in terms of a *lift curve,* which plots the CA rate as a function of the total fraction of subscribers identified by the model. Problem (1) thus imposes the constraint that the solution lies at one coordinate of the lift curve, just as problems (2) and (3) place the constraint that the solution lies at one coordinate of the ROC curve. Thus, a solution to problems (2) or (3) will also serve as a solution to (1). Although addressing problem (4) seems most fanciful, it encompasses problems (2) and (3), and thus we focus on it. Our goal is not altogether unreasonable, because a solution to problem

(4) has the property we characterized in Figure 1b: the ROC curve can suffer everywhere except in the region near CA $\alpha_{CA}$ and CR $\alpha_{CR}$. Hence, the approaches we consider will trade off performance in some regions of the ROC curve against performance in other regions. We call this *prodding* the ROC curve.

## 2 FOUR ALGORITHMS TO PROD THE ROC CURVE

In this section, we describe four algorithms for prodding the ROC curve toward a target CA rate of $\alpha_{CA}$ and a target CR rate of $\alpha_{CR}$.

### 2.1 EMPHASIZING CRITICAL TRAINING EXAMPLES

Suppose we train a classifier on a set of positive and negative examples from a class—churners and nonchurners in our domain. Following training, the classifier will assign a posterior probability of class membership to each example. The examples can be sorted by the posterior and arranged on a continuum anchored by probabilities 0 and 1 (Figure 2). We can identify the thresholds, $\theta_{CA}$ and $\theta_{CR}$, which yield CA and CR rates of $\alpha_{CA}$ and $\alpha_{CR}$, respectively. If the classifier's discrimination performance fails to achieve the target CA and CR rates, then $\theta_{CA}$ will be lower than $\theta_{CR}$, as depicted in the Figure. If we can bring these two thresholds together, we will achieve the target CA and CR rates. Thus, the first algorithm we propose involves training a series of classifiers, attempting to make classifier $n+1$ achieve better CA and CR rates by focusing its effort on examples from classifier $n$ that lie between $\theta_{CA}$ and $\theta_{CR}$; the positive examples must be pushed above $\theta_{CR}$ and the negative examples must be pushed below $\theta_{CA}$. (Of course, the thresholds are specific to a classifier, and hence should be indexed by $n$.) We call this the *emphasis* algorithm, because it involves placing greater weight on the examples that lie between the two thresholds. In the Figure, the emphasis for classifier $n+1$ would be on examples $e_5$ through $e_8$. This retraining procedure can be iterated until the classifier's training set performance reaches asymptote.

In our implementation, we define a weighting of each example $i$ for training classifier $n$, $\lambda_i^n$. For classifier 1, $\lambda_i^1 = 1$. For subsequent classifiers, $\lambda_i^{n+1} = \lambda_i^n$ if example $i$ is not in the region of emphasis, or $\lambda_i^{n+1} = \kappa_e \lambda_i^n$ otherwise, where $\kappa_e$ is a constant, $\kappa_e > 1$.

### 2.2 DEEMPHASIZING IRRELEVANT TRAINING EXAMPLES

The second algorithm we propose is related to the first, but takes a slightly different perspective on the continuum depicted in Figure 2. Positive examples below $\theta_{CA}$—such as $e_2$—are clearly the most difficult positive examples to classify correctly. Not only are they the most difficult positive examples, but they do not in fact need to be classified correctly to achieve the target CA and CR rates. Threshold $\theta_{CR}$ does not depend on examples such as $e_2$, and threshold $\theta_{CA}$ allows a fraction $(1-\alpha_{CA})$ of the positive examples to be classified incorrectly. Likewise, one can argue that negative examples above $\theta_{CR}$—such as $e_{10}$ and $e_{11}$—need not be of concern. Essentially, the second algorithm, which we term the *deemphasis* algorithm, is like the emphasis algorithm in that a series of classifiers are trained, but when training classifier $n+1$, *less* weight is placed on the examples whose correct clas-

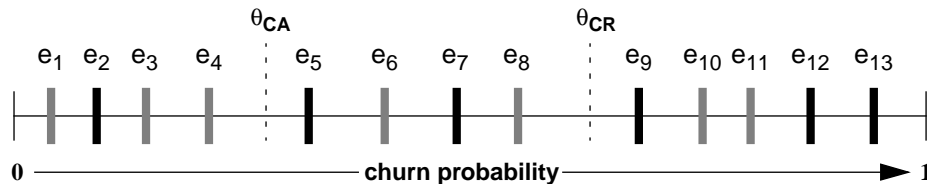

**FIGURE 2. A schematic depiction of all training examples arranged by the classifier's posterior. Each solid bar corresponds to a positive example (e.g., a churner) and each grey bar corresponds to a negative example (e.g., a nonchurner).**

sification is *unnecessary* to achieve the target CA and CR rates for classifier $n$. As with the emphasis algorithm, the retraining procedure can be iterated until no further performance improvements are obtained on the training set. Note that the set of examples given emphasis by the previous algorithm is *not* the complement of the set of examples deemphasized by the current algorithm; the algorithms are *not* identical.

In our implementation, we assign a weight to each example $i$ for training classifier $n$, $\lambda_i^n$. For classifier 1, $\lambda_i^1 = 1$. For subsequent classifiers, $\lambda_i^{n+1} = \lambda_i^n$ if example $i$ is not in the region of deemphasis, or $\lambda_i^{n+1} = \kappa_d \lambda_i^n$ otherwise, where $\kappa_d$ is a constant, $\kappa_d < 1$.

## 2.3 CONSTRAINED OPTIMIZATION

The third algorithm we propose is formulated as maximizing the CR rate while maintaining the CA rate equal to $\alpha_{CA}$. (We do not attempt to simultaneously maximize the CA rate while maintaining the CR rate equal to $\alpha_{CR}$.) Gradient methods cannot be applied directly because the CA and CR rates are nondifferentiable, but we can approximate the CA and CR rates with smooth differentiable functions:

$$\overline{CA}(w, t) = \frac{1}{|P|} \sum_{i \in P} \sigma_\beta(f(x_i, w) - t) \qquad \overline{CR}(w, t) = \frac{1}{|N|} \sum_{i \in N} \sigma_\beta(t - f(x_i, w)),$$

where $P$ and $N$ are the set of positive and negative examples, respectively, $f(x,w)$ is the model posterior for input $x$, $w$ is the parameterization of the model, $t$ is a threshold, and $\sigma_\beta$ is a sigmoid function with scaling parameter $\beta$: $\sigma_\beta(y) = (1 + \exp(-\beta y))^{-1}$. The larger $\beta$ is, the more nearly step-like the sigmoid is and the more nearly equal the approximations are to the model CR and CA rates. We consider the problem formulation in which $\overline{CA}$ is a constraint and $\overline{CR}$ is a figure of merit. We convert the constrained optimization problem into an unconstrained problem by the augmented Lagrangian method (Bertsekas, 1982), which involves iteratively maximizing an objective function

$$A(w, t) = \overline{CR}(w, t) + \nu \left| \overline{CA}(w, t) - \alpha_{CA} \right| + \frac{\mu}{2} \left| \overline{CA}(w, t) - \alpha_{CA} \right|^2$$

with a fixed Lagrangian multiplier, $\nu$, and then updating $\nu$ following the optimization step: $\nu \leftarrow \nu + \mu \left| \overline{CA}(w^*, t^*) - \alpha_{CA} \right|$, where $w^*$ and $t^*$ are the values found by the optimization step. We initialize $\nu = 1$ and fix $\mu = 1$ and $\beta = 10$ and iterate until $\nu$ converges.

## 2.4 GENETIC ALGORITHM

The fourth algorithm we explore is a steady-state genetic search over a space defined by the continuous parameters of a classifier (Whitley, 1989). The fitness of a classifier is the reciprocal of the number of training examples falling between the $\theta_{CA}$ and $\theta_{CR}$ thresholds. Much like the emphasis algorithm, this fitness function encourages the two thresholds to come together. The genetic search permits direct optimization over a nondifferentiable criterion, and therefore seems sensible for the present task.

# 3 METHODOLOGY

For our tests, we studied two large data bases made available to Athene by two telecommunications providers. Data set 1 had 50,000 subscribers described by 35 input features and a churn rate of 4.86%. Data set 2 had 169,727 subscribers described by 51 input features and a churn rate of 6.42%. For each data base, the features input to the classifier were obtained by proprietary transformations of the raw data (see Mozer et al., 2000). We chose these two large, real world data sets because achieving gains with these data sets should be more difficult than with smaller, less noisy data sets. Plus, with our real-world data, we can evaluate the cost savings achieved by an improvement in prediction accuracy. We performed 10-fold cross-validation on each data set, preserving the overall churn/nonchurn ratio in each split.

In all tests, we chose $\alpha_{CR} = 0.90$ and $\alpha_{CA} = 0.50$, values which, based on our past experience in this domain, are ambitious yet realizable targets for data sets such as

these. We used a logistic regression model (i.e., a no hidden unit neural network) for our studies, believing that it would be more difficult to obtain improvements with such a model than with a more flexible multilayer perceptron. For the emphasis and deemphasis algorithms, models were trained to minimize mean-squared error on the training set. We chose $\kappa_e = 1.3$ and $\kappa_d = .75$ by quick exploration. Because the weightings are cumulative over training restarts, the choice of $\kappa$ is not critical for either algorithm; rather, the magnitude of $\kappa$ controls how many restarts are necessary to reach asymptotic performance, but the results we obtained were robust to the choice of $\kappa$. The emphasis and deemphasis algorithms were run for 100 iterations, which was the number of iterations required to reach asymptotic performance on the training set.

## 4  RESULTS

Figure 3 illustrates training set performance for the emphasis algorithm on data set 1. The graph on the left shows the CA rate when the CR rate is .9, and the graph on the right show the CR rate when the CA rate is .5. Clearly, the algorithm appears to be stable, and the ROC curve is improving in the region around ($\alpha_{CA}$, $\alpha_{CR}$).

Figure 4 shows cross-validation performance on the two data sets for the four prodding algorithms as well as for a traditional least-squares training procedure. The emphasis and deemphasis algorithms yield reliable improvements in performance in the critical region of the ROC curve over the traditional training procedure. The constrained-optimization and genetic algorithms perform well on achieving a high CR rate for a fixed CA rate, but neither does as well on achieving a high CA rate for a fixed CR rate. For the constrained-optimization algorithm, this result is not surprising as it was trained asymmetrically, with the CA rate as the constraint. However, for the genetic algorithm, we have little explanation for its poor performance, other than the difficulty faced in searching a continuous space without gradient information.

## 5  DISCUSSION

In this paper, we have identified an interesting, novel problem in classifier design which is motivated by our domain of churn prediction and real-world business considerations. Rather than seeking a classifier that maximizes discriminability between two classes, as measured by area under the ROC curve, we are concerned with optimizing performance at certain points along the ROC curve. We presented four alternative approaches to prodding the ROC curve, and found that all four have promise, depending on the specific goal.

Although the magnitude of the gain is small—an increase of about .01 in the CR rate given a target CA rate of .50—the impro vement results in significant dollar savings. Using a framework for evaluating dollar savings to a service provider, based on estimates of subscriber retention and costs of intervention obtained in real world data collection (Mozer et

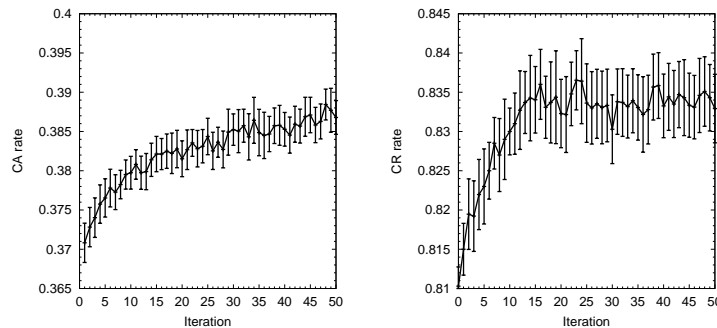

**FIGURE 3. Training set performance for the emphasis algorithm on data set 1. (a) CA rate as a function of iteration for a CR rate of .9; (b) CR rate as a function of iteration for a CA rate of .5. Error bars indicate +/–1 standard error of the mean.**

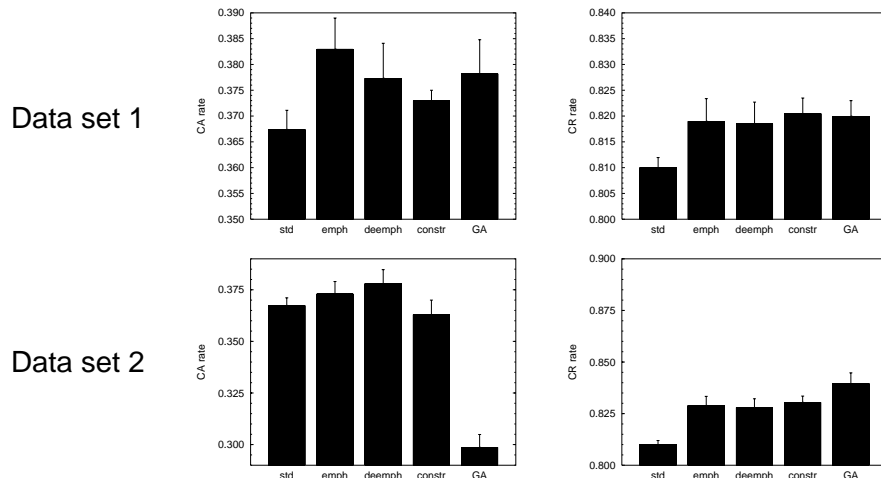

**FIGURE 4. Cross-validation performance on the two data sets for the standard training procedure (STD), as well as the emphasis (EMPH), deemphasis (DEEMPH), constrained optimization (CONSTR), and genetic (GEN) algorithms. The left column shows the CA rate for CR rate .9; the right column shows the CR rate for CA rate .5. The error bar indicates one standard error of the mean over the 10 data splits.**

al., 2000), we obtain a savings of $11 per churnable subscriber when the (CA, CR) rates go from (.50, .80) to (.50, .81), which amounts to an 8% increase in profitability of the subscriber intervention effort.

These figures are clearly promising. However, based on the data sets we have studied, it is difficult to know whether another algorithm might exist that achieves even greater gains. Interestingly, all algorithms we proposed yielded roughly the same gains when successful, suggesting that we may have milked the data for whatever gain could be had, given the model class evaluated. Our work clearly illustrate the difficulty of the problem, and we hope that others in the NIPS community will be motivated by the problem to suggest even more powerful, theoretically grounded approaches.

## 6 ACKNOWLEDGEMENTS

No white males were angered in the course of conducting this research. We thank Lian Yan and David Grimes for comments and assistance on this research. This research was supported in part by McDonnell-Pew grant 97-18, NSF award IBN-9873492, and NIH/IFOPAL R01 MH61549–01A1.

## 7 REFERENCES

Bertsekas, D. P. (1982). *Constrained optimization and Lagrange multiplier methods*. NY: Academic.

Chang, E. I., & Lippmann, R. P. (1994). Figure of merit training for detection and spotting. In J. D. Cowan, G. Tesauro, & J. Alspector (Eds.), *Advances in Neural Information Processing Systems 6* (1019–1026). San Mateo, CA: Morgan Kaufmann.

Frederick, E. D., & Floyd, C. E. (1998). Analysis of mammographic findings and patient history data with genetic algorithms for the prediction of breast cancer biopsy outcome. *Proceedings of the SPIE, 3338,* 241–245.

Green, D. M., & Swets, J. A. (1966). *Signal detection theory and psychophysics*. New York: Wiley.

Mozer, M. C., Wolniewicz, R., Grimes, D., Johnson, E., & Kaushansky, H. (2000). Maximizing revenue by predicting and addressing customer dissatisfaction. *IEEE Transactions on Neural Networks*, *11*, 690–696.

Whitley, D. (1989). The GENITOR algorithm and selective pressure: Why rank-based allocation of reproductive trials is best. In D. Schaffer (Ed.), Proceedings of the Third International Conference on Genetic Algorithms (pp. 116–121). San Mateo, CA: Morgan Kaufmann.